# Feudal Reinforcement Learning

**Peter Dayan**
CNL
The Salk Institute
PO Box 85800
San Diego CA 92186-5800, USA
dayan@helmholtz.sdsc.edu

**Geoffrey E Hinton**
Department of Computer Science
University of Toronto
6 Kings College Road, Toronto,
Canada M5S 1A4
hinton@ai.toronto.edu

## Abstract

One way to speed up reinforcement learning is to enable learning to happen simultaneously at multiple resolutions in space and time. This paper shows how to create a $Q$-learning managerial hierarchy in which high level managers learn how to set tasks to their sub-managers who, in turn, learn how to satisfy them. Sub-managers need not initially understand their managers' commands. They simply learn to maximise their reinforcement in the context of the current command.

We illustrate the system using a simple maze task.. As the system learns how to get around, satisfying commands at the multiple levels, it explores more efficiently than standard, flat, $Q$-learning and builds a more comprehensive map.

## 1  INTRODUCTION

Straightforward reinforcement learning has been quite successful at some relatively complex tasks like playing backgammon (Tesauro, 1992). However, the learning time does not scale well with the number of parameters. For agents solving rewarded Markovian decision tasks by learning dynamic programming value functions, some of the main bottlenecks (Singh, 1992b) are *temporal resolution* − expanding the unit of learning from the smallest possible step in the task, *division-and-conquest* − finding smaller subtasks that are easier to solve, *exploration*, and *structural generalisation* − generalisation of the value function between different lo-

cations. These are obviously related – for instance, altering the temporal resolution can have a dramatic effect on exploration.

Consider a control hierarchy in which managers have sub-managers, who work for them, and super-managers, for whom they work. If the hierarchy is *strict* in the sense that managers control exactly the sub-managers at the level below them and only the very lowest level managers can actually act in the world, then intermediate level managers have essentially two instruments of control over their sub-managers at any time – they can choose amongst them and they can set them sub-tasks. These sub-tasks can be incorporated into the *state* of the sub-managers so that they in turn can choose their own sub-sub-tasks and sub-sub-managers to execute them based on the task selection at the higher level.

An appropriate hierarchy can address the first three bottlenecks. Higher level managers should sustain a larger grain of temporal resolution, since they leave the sub-sub-managers to do the actual work. Exploration for actions leading to rewards can be more efficient since it can be done non-uniformly – high level managers can decide that reward is best found in some other region of the state space and send the agent there directly, without forcing it to explore in detail on the way.

Singh (1992a) has studied the case in which a manager picks one of its sub-managers rather than setting tasks. He used the degree of accuracy of the $Q$-values of sub-managerial $Q$-learners (Watkins, 1989) to train a gating system (Jacobs, Jordan, Nowlan & Hinton, 1991) to choose the one that matches best in each state. Here we study the converse case, in which there is only one possible sub-manager active at any level, and so the only choice a manager has is over the tasks it sets. Such systems have been previously considered (Hinton, 1987; Watkins, 1989).

The next section considers how such a strict hierarchical scheme can learn to choose appropriate tasks at each level, section 3 describes a maze learning example for which the hierarchy emerges naturally as a multi-grid division of the space in which the agent moves, and section 4 draws some conclusions.

## 2   FEUDAL CONTROL

We sought to build a system that mirrored the hierarchical aspects of a feudal fiefdom, since this is one extreme for models of control. Managers are given absolute power over their sub-managers – they can set them tasks and reward and punish them entirely as they see fit. However managers ultimately have to satisfy their own super-managers, or face punishment themselves – and so there is recursive reinforcement and selection until the whole system satisfies the goal of the highest level manager. This can all be made to happen without the sub-managers initially "understanding" the sub-tasks they are set. Every component just acts to maximise its expected reinforcement, so after learning, the meaning it attaches to a specification of a sub-task consists of the way in which that specification influences its choice of sub-sub-managers and sub-sub-tasks. Two principles are key:

**Reward Hiding** Managers must reward sub-managers for doing their bidding *whether or not* this satisfies the commands of the super-managers. Sub-managers should just learn to obey their managers and leave it up to them to determine what

it is best to do at the next level up. So if a sub-manager fails to achieve the sub-goal set by its manager it is not rewarded, even if its actions result in the satisfaction of of the manager's own goal. Conversely, if a sub-manager achieves the sub-goal it is given it is rewarded, even if this does not lead to satisfaction of the manager's own goal. This allows the sub-manager to learn to achieve sub-goals even when the manager was mistaken in setting these sub-goals. So in the early stages of learning, low-level managers can become quite competent at achieving low-level goals even if the highest level goal has never been satisfied.

**Information Hiding** Managers only need to know the state of the system at the granularity of their own choices of tasks. Indeed, allowing some decision making to take place at a coarser grain is one of the main goals of the hierarchical decomposition. Information is hidden both downwards – sub-managers do not know the task the super-manager has set the manager – and upwards – a super-manager does not know what choices its manager has made to satisfy its command. However managers do need to know the satisfaction conditions for the tasks they set and some measure of the actual cost to the system for achieving them using the sub-managers and tasks it picked on any particular occasion.

For the special case to be considered here, in which managers are given no choice of which sub-manager to use in a given state, their choice of a task is very similar to that of an action for a standard $Q$-learning system. If the task is completed successfully, the cost is determined by the super-manager according to how well (*eg* how quickly, or indeed whether) the manager satisfied its super-tasks. Depending on how its own task is accomplished, the manager rewards or punishes the sub-manager responsible. When a manager chooses an action, control is passed to the sub-manager and is only returned when the state changes *at the managerial level.*

## 3  THE MAZE TASK

To illustrate this feudal system, consider a standard maze task (Barto, Sutton & Watkins, 1989) in which the agent has to learn to find an initially unknown goal. The grid is split up at successively finer grains (see figure 1) and managers are assigned to separable parts of the maze at each level. So, for instance, the level 1 manager of area 1-(1,1) sets the tasks for and reinforcement given to the level 2 managers for areas 2-(1,1), 2-(1,2), 2-(2,1) and 2-(2,2). The successive separation into quarters is fairly arbitrary – however if the regions at high levels did not cover contiguous areas at lower levels, then the system would not perform very well.

At all times, the agent is effectively performing an action at every level. There are five actions, NSEW and *, available to the managers at all levels other than the first and last. NSEW represent the standard geographical moves and * is a special action that non-hierarchical systems do not require. It specifies that lower level managers should search for the goal within the confines of the current larger state instead of trying to move to another region of the space at the same level. At the top level, the only possible action is *; at the lowest level, only the geographical moves are allowed, since the agent cannot search at a finer granularity than it can move.

Each manager maintains $Q$ values (Watkins, 1989; Barto, Bradtke & Singh, 1992) over the actions it instructs its sub-managers to perform, based on the location of

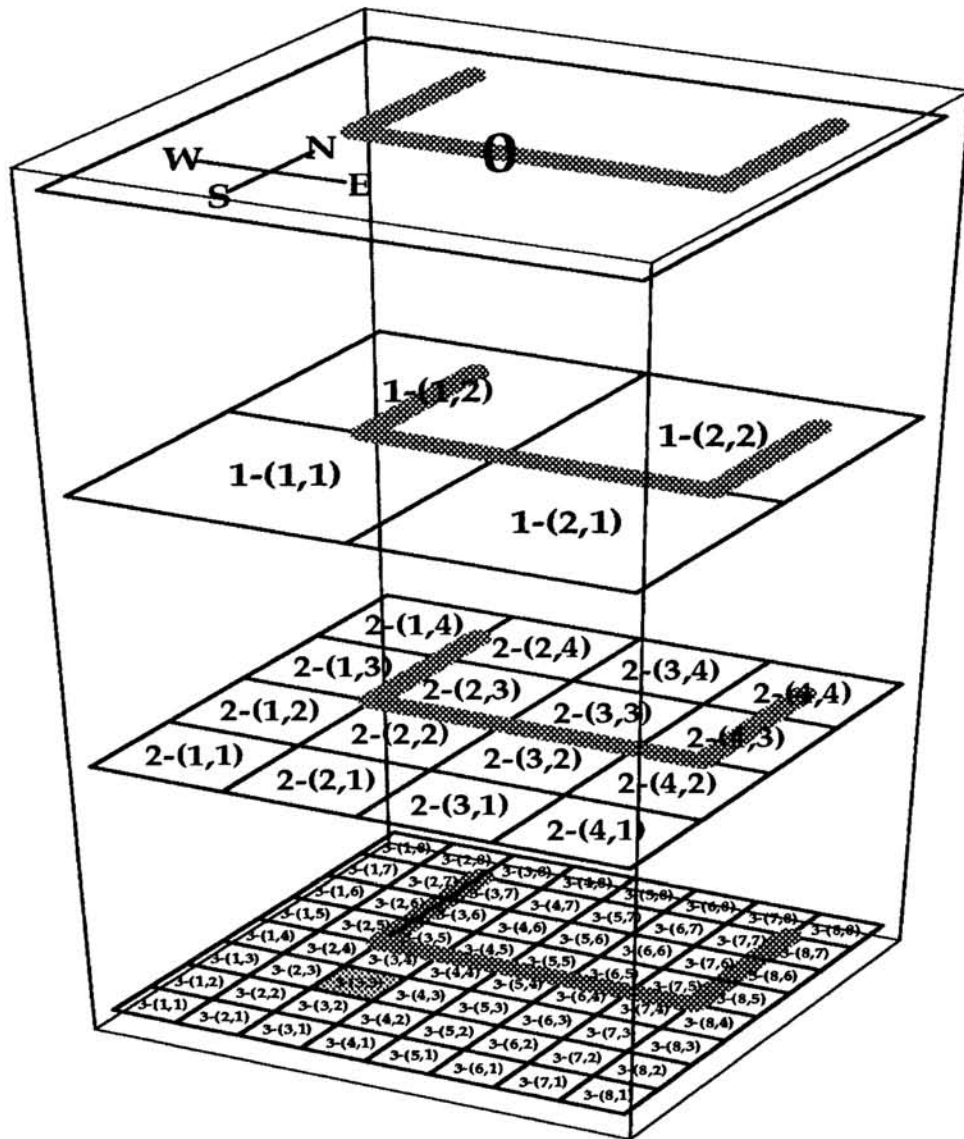

Figure 1: The Grid Task. This shows how the maze is divided up at different levels in the hierarchy. The 'U' shape is the barrier, and the shaded square is the goal. Each high level state is divided into four low level ones at every step.

the agent at the subordinate level of detail and the command it has received from above. So, for instance, if the agent currently occupies 3-(6,6), and the instruction from the level 0 manager is to move South, then the 1-(2,2) manager decides upon an action based on the $Q$ values for NSEW giving the total length of the path to either 2-(3,2) or 2-(4,2). The action the 1-(2,2) manager chooses is communicated one level down the hierarchy and becomes part of the state determining the level 2 $Q$ values.

When the agent starts, actions at successively lower levels are selected using the standard $Q$-learning softmax method and the agent moves according to the finest grain action (at level 3 here). The $Q$ values at every level at which this causes

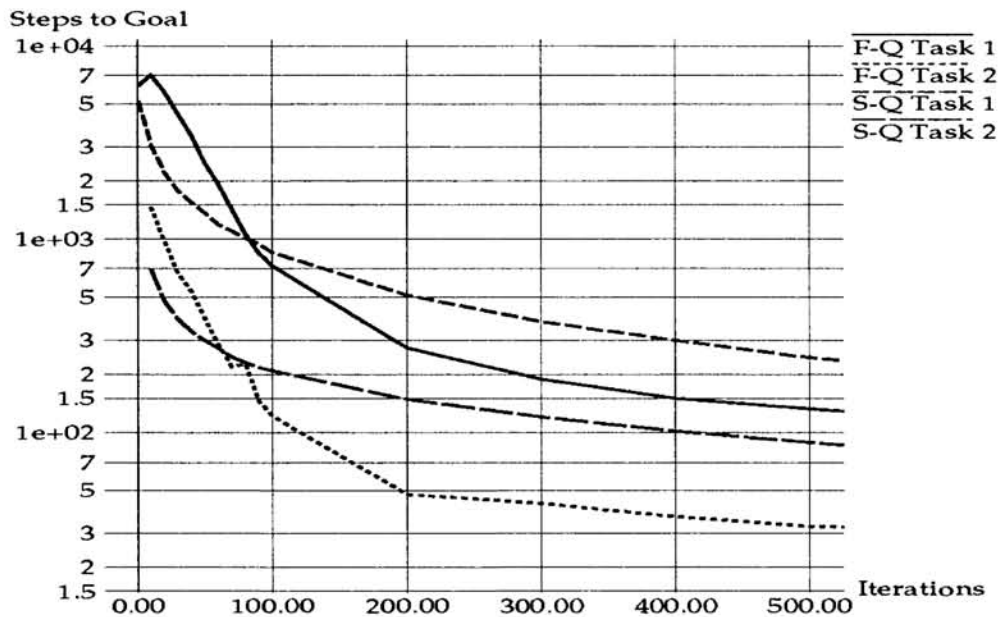

Figure 2: Learning Performance. F-Q shows the performance of the feudal architecture and S-Q of the standard $Q$-learning architecture.

a state transition are updated according to the length of path *at that level*, if the state transition is what was ordered at all lower levels. This restriction comes from the constraint that super-managers should only learn from the fruits of the honest labour of sub-managers, *ie* only if they obey their managers.

Figure 2 shows how the system performs compared with standard, one-step, $Q$-learning, first in finding a goal in a maze similar to that in figure 1, only having 32x32 squares, and second in finding the goal after it is subsequently moved. Points on the graph are averages of the number of steps it takes the agent to reach the goal across all possible testing locations, after the given number of learning iterations. Little effort was made to optimise the learning parameters, so care is necessary in interpreting the results.

For the first task the feudal system is initially slower, but after a while, it learns much more quickly how to navigate to the goal. The early sloth is due to the fact that many low level actions are wasted, since they do not implement desired higher level behaviour and the system has to learn not to try impossible actions or * in inappropriate places. The late speed comes from the feudal system's superior exploratory behaviour. If it decides at a high level that the goal is in one part of the maze, then it has the capacity to specify large scale actions at that level to take it there. This is the same advantage that Singh's (1992b) variable temporal resolution system garners, although this is over a single task rather than explicitly composite sub-tasks. Tests on mazes of different sizes suggested that the number of iterations after which the advantage of exploration outweighs the disadvantage of wasted actions gets less as the complexity of the task increases.

A similar pattern emerges for the second task. Low level $Q$ values embody an implicit knowledge of how to get around the maze, and so the feudal system can explore efficiently once it (slowly) learns not to search in the original place.

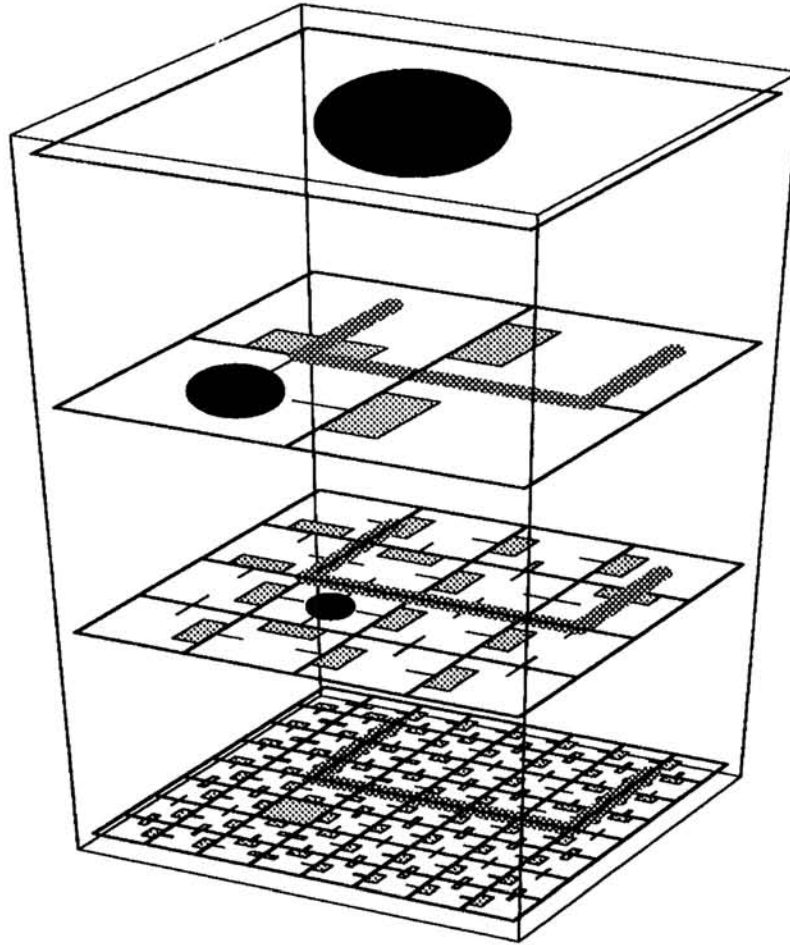

Figure 3: The Learned Actions. The area of the boxes and the radius of the central circle give the probabilities of taking action NSEW and * respectively.

---

Figure 3 shows the probabilities of each move at each location once the agent has learnt to find the goal at 3-(3,3). The length of the NSEW bars and the radius of the central circle are proportional to the probability of selecting actions NSEW or * respectively, and action choice flows from top to bottom. For instance, the probability of choosing action S at state 2-(1,3) is the sum of the products of the probabilities of choosing actions NSEW and * at state 1-(1,2) and the probabilities, conditional on this higher level selection, of choosing action S at state 2-(1,3). Apart from the right hand side of the barrier, the actions are generally correct – however there are examples of sub-optimal behaviour caused by the decomposition of the space, *eg* the system decides to move North at 3-(8,5) despite it being more felicitous to move South.

Closer investigation of the course of learning reveals that, as might be expected from the restrictions in updating the $Q$ values, the system initially learns in a completely bottom-up manner. However after a while, it learns appropriate actions at the highest levels, and so top-down learning happens too. This generally beneficial effect arises because there are far fewer states at coarse resolutions, and so it is easier for the agent to calculate what to do.

# 4  DISCUSSION

The feudal architecture partially addresses one of the major concerns in reinforcement learning about how to divide a single task up into sub-tasks at multiple levels. A demonstration was given of how this can be done separately from choosing between different possible sub-managers at a given level.

It depends on there being a plausible managerial system, preferably based on a natural hierarchical division of the available state space. For some tasks it can be very inefficient, since it forces each sub-manager to learn how to satisfy all the sub-tasks set by its manager, whether or not those sub-tasks are appropriate. It is therefore more likely to be useful in environments in which the set tasks can change. Managers need not necessarily know in advance the consequences of their actions. They could learn, in a self-supervised manner, information about the state transitions that they have experienced. These observed next states can be used as goals for their sub-managers – consistency in providing rewards for appropriate transitions is the only requirement.

Although the system gains power through hiding information, which reduces the size of the state spaces that must be searched, such a step also introduces inefficiencies. In some cases, if a sub-manager only knew the super-task of its super-manager then it could bypass its manager with advantage. However the *reductio* of this would lead to each sub-manager having as large a state space as the whole problem, negating the intent of the feudal architecture. A more serious concern is that the non-Markovian nature of the task at the higher levels (the future course of the agent is determined by more detailed information than just the high level states) can render the problem insoluble. Moore and Atkeson's (1993) system for detecting such cases and choosing finer resolutions accordingly should integrate well with the feudal system.

For the maze task, the feudal system learns much more about how to navigate than the standard $Q$-learning system. Whereas the latter is completely concentrated on a particular target, the former knows how to execute arbitrary high level moves efficiently, even ones that are not used to find the current goal such as going East from one quarter of the space 1-(2,2) to another 1-(1,2). This is why exploration can be more efficient. It doesn't require a map of the space, or even a model of `state x action` → `next state` to be learned explicitly.

Jameson (1992) independently studied a system with some similarities to the feudal architecture. In one case, a high level agent learned on the basis of external reinforcement to provide on a slow timescale direct commands (like reference trajectories) to a low level agent – which learned to obey it based on reinforcement proportional to the square trajectory error. In another, low and high level agents received the same reinforcement from the world, but the former was additionally tasked on making its prediction of future reinforcement significantly dependent on the output of the latter. Both systems learned very effectively to balance an upended pole for long periods. They share the notion of hierarchical structure with the feudal architecture, but the notion of control is somewhat different.

Multi-resolution methods have long been studied as ways of speeding up dynamic programming (see Morin, 1978, for numerous examples and references). Standard

methods focus effectively on having a single task at every level and just having coarser and finer representations of the value function. However, here we have studied a slightly different problem in which managers have the flexibility to specify different tasks which the sub-managers have to learn how to satisfy. This is more complicated, but also more powerful.

From a psychological perspective, we have replaced a system in which there is a single external reinforcement schedule with a system in which the rat's mind is composed of a hierarchy of little Skinners.

## Acknowledgements

We are most grateful to Andrew Moore, Mark Ring, Jürgen Schmidhuber, Satinder Singh, Sebastian Thrun and Ron Williams for helpful discussions. This work was supported by SERC, the Howard Hughes Medical Institute and the Canadian Institute for Advanced Research (CIAR). GEH is the Noranda fellow of the CIAR.

# References

[1] Barto, AG, Bradtke, SJ & Singh, SP (1991). *Real-Time Learning and Control using Asynchronous Dynamic Programming*. COINS technical report 91-57. Amherst: University of Massachusetts.

[2] Barto, AG, Sutton, RS & Watkins, CJCH (1989). Learning and sequential decision making. In M Gabriel & J Moore, editors, *Learning and Computational Neuroscience: Foundations of Adaptive Networks*. Cambridge, MA: MIT Press, Bradford Books.

[3] Hinton, GE (1987). *Connectionist Learning Procedures*. Technical Report CMU-CS-87-115, Department of Computer Science, Carnegie-Mellon University.

[4] Jacobs, RA, Jordan, MI, Nowlan, SJ & Hinton, GE. Adaptive mixtures of local experts. *Neural Computation*, **3**, pp 79-87.

[5] Jameson, JW (1992). Reinforcement control with hierarchical backpropagated adaptive critics. Submitted to *Neural Networks*.

[6] Moore, AW & Atkeson, CG (1993). Memory-based reinforcement learning: efficient computation with prioritized sweeping. In SJ Hanson, CL Giles & JD Cowan, editors *Advances in Neural Information Processing Systems 5*. San Mateo, CA: Morgan Kaufmann.

[7] Morin, TL (1978). Computational advances in dynamic programming. In ML Puterman, editor, *Dynamic Programming and its Applications*. New York: Academic Press.

[8] Moore, AW (1991). Variable resolution dynamic programming: Efficiently learning action maps in multivariate real-valued state spaces. *Proceedings of the Eighth Machine Learning Workshop*. San Mateo, CA: Morgan Kaufmann.

[9] Singh, SP (1992a). Transfer of learning by composing solutions for elemental sequential tasks. *Machine Learning*, **8**, pp 323-340.

[10] Singh, SP (1992b). Scaling reinforcement learning algorithms by learning variable temporal resolution models. Submitted to *Machine Learning*.

[11] Tesauro, G (1992). Practical issues in temporal difference learning. *Machine Learning*, **8**, pp 257-278.

[12] Watkins, CJCH (1989). *Learning from Delayed Rewards*. PhD Thesis. University of Cambridge, England.